# Learning Anchor Planes for Classification

**Ziming Zhang**[†]    **Ľubor Ladický**[‡]    **Philip H.S. Torr**[†]    **Amir Saffari**[†§]

† Department of Computing, Oxford Brookes University, Wheatley, Oxford, OX33 1HX, U.K.
‡ Department of Engineering Science, University of Oxford, Parks Road, Oxford, OX1 3PJ, U.K.
§ Sony Computer Entertainment Europe, London, UK
{ziming.zhang, philiptorr}@brookes.ac.uk
lubor@robots.ox.ac.uk
amir@ymer.org

## Abstract

Local Coordinate Coding (LCC) [18] is a method for modeling functions of data lying on non-linear manifolds. It provides a set of anchor points which form a local coordinate system, such that each data point on the manifold can be approximated by a linear combination of its anchor points, and the linear weights become the local coordinate coding. In this paper we propose encoding data using orthogonal anchor planes, rather than anchor points. Our method needs only a few orthogonal anchor planes for coding, and it can linearize any $(\alpha, \beta, p)$-Lipschitz smooth non-linear function with a fixed expected value of the upper-bound approximation error on any high dimensional data. In practice, the orthogonal coordinate system can be easily learned by minimizing this upper bound using singular value decomposition (SVD). We apply our method to model the coordinates locally in linear SVMs for classification tasks, and our experiment on MNIST shows that using only 50 anchor planes our method achieves 1.72% error rate, while LCC achieves 1.90% error rate using 4096 anchor points.

## 1   Introduction

Local Coordinate Coding (LCC) [18] is a coding scheme that encodes the data locally so that any non-linear $(\alpha, \beta, p)$-Lipschitz smooth function (see Definition 1 in Section 2 for details) over the data manifold can be approximated using linear functions. There are two components in this method: (1) a set of anchor points which decide the local coordinates, and (2) the coding for each data based on the local coordinates given the anchor points. Theoretically [18] suggests that under certain assumptions, locality is more essential than sparsity for non-linear function approximation. LCC has been successfully applied to many applications such like object recognition (*e.g.* locality-constraint linear coding (LLC) [16]) in VOC 2009 challenge [7].

One big issue in LCC is that its classification performance is highly dependent on the number of anchor points, as observed in Yu and Zhang [19], because these points should be "local enough" to encode surrounding data on the data manifold accurately, which sometimes means that in real applications the number of anchor points explodes to a surprisingly huge number. This has been demonstrated in [18] where LCC has been tested on MNIST dataset, using from 512 to 4096 anchor points learned from sparse coding, the error rate decreased from 2.64% to 1.90%. This situation could become a serious problem when the distribution of the data points is sparse in the feature space, *i.e.* there are many "holes" between data points (*e.g.* regions of feature space that are sparsely populated by data). As a result of this, many redundant anchor points will be distributed in the holes with little information. By using many anchor points, the computational complexity of the classifier at both training and test time increases significantly, defeating the original purpose of using LCC.

So far several approaches have been proposed for problems closely related to anchor point learning such as dictionary learning or codebook learning. For instance, Lee *et. al.* [12] proposed learning the anchor points for sparse coding using the Lagrange dual. Mairal *et. al.* [13] proposed an online dictionary learning algorithm using stochastic approximations. Wang *et. al.* [16] proposed locality-constraint linear coding (LLC), which is a fast implementation of LCC, and an online incremental codebook learning algorithm using coordinate descent method, whose performance is very close to that using K-Means. However, none of these algorithms can deal with holes of sparse data as they need many anchor points.

In this paper, we propose a method to approximate any non-linear $(\alpha, \beta, p)$-Lipschitz smooth function using an orthogonal coordinate coding (OCC) scheme on a set of orthogonal basis vectors. Each basis vector $\mathbf{v} \in \mathbb{R}^d$ defines a family of **anchor planes**, each of which can be considered as consisting of infinite number of anchor points, and the nearest point on each anchor plane to a data point $\mathbf{x} \in \mathbb{R}^d$ is used for coding, as illustrated in Figure 1. The data point $\mathbf{x}$ will be encoded based on the margin, $\mathbf{x}^T \mathbf{v}$ where $(\cdot)^T$ denotes the matrix transpose operator, between $\mathbf{x}$ and an anchor plane defined by $\mathbf{v}$. The benefits of using anchor planes are:

- A few anchor planes can replace many anchor points while preserving similar locality of anchor points. This sparsity may lead to a better generalization since many anchor points will overfit the data easily. Therefore, it can deal with the hole problem in LCC.

- The learned orthogonal basis vectors can fit naturally into locally linear SVM's (such as [9,10,11,19,21]) which we describe below.

Theoretically we show that using OCC any $(\alpha, \beta, p)$-Lipschitz smooth non-linear function can be linearized with a fixed upper-bound approximation error. In practice by minimizing this upper bound, the orthogonal basis vectors can be learned using singular value decomposition (SVD). In our experiments, We integrate OCC into LL-SVM for classification.

Linear support vector machines have become popular for solving classification tasks due to their fast and simple online application to large scale data sets. However, many problems are not linearly separable. For these problems kernel-based SVMs are often used, but unlike their linear variant they suffer from various drawbacks in terms of computational and memory efficiency. Their response can be represented only as a function of the set of support vectors, which has been experimentally shown to grow linearly with the size of the training set. A recent trend has grown to create a classifier locally based on a set of linear SVMs [9,10,11,19,21]. For instance, in [20] SVMs are trained only based on the N nearest neighbors of each data, and in [9] multiple kernel learning was applied locally. In [10] Kecman and Brooks proved that the stability bounds for local SVMs are tighter than the ones for traditional, global, SVMs. Ladicky and Torr [11] proposed a novel locally linear SVM classifier (LL-SVM) with smooth decision boundary and bounded curvature. They show how the functions defining the classifier can be approximated using local codings and show how this model can be optimized in an online fashion by performing stochastic gradient descent with the same convergence guarantees as standard gradient descent method for linear SVMs. Mathematically LL-SVM is formulated as follows:

$$\arg\min_{\mathbf{W}, \mathbf{b}} \quad \frac{\lambda}{2} \|\mathbf{W}\|^2 + \frac{1}{|\mathcal{S}|} \sum_{k \in \mathcal{S}} \xi_k \qquad (1)$$
$$\text{s.t.} \quad \forall k \in \mathcal{S}: \quad \xi_k \geq 1 - y_k \left[ \boldsymbol{\gamma}_{x_k}^T \mathbf{W} \mathbf{x}_k + \boldsymbol{\gamma}_{x_k}^T \mathbf{b} \right], \xi_k \geq 0$$

where $\forall k$, $\mathbf{x}_k \in \mathbb{R}^d$ is a training vector, $y_k \in \{-1, 1\}$ is its label, $\boldsymbol{\gamma}_{x_k} \in \mathbb{R}^N$ is its local coding, $\lambda \geq 0$ is a pre-defined scalar, and $\mathbf{W} \in \mathbb{R}^{N \times d}$ and $\mathbf{b} \in \mathbb{R}^N$ are the model parameters. As demonstrated in our experiments, the choices of the local coding methods are very important for LL-SVM, and an improper choice will hurt its performance.

The rest of the paper is organized as follows. In Section 2 we first recall some definitions and lemmas in LCC, then introduce OCC for non-linear function approximation and its property on the upper bound of localization error as well as comparing OCC with LCC in terms of geometric interpretation and optimization. In Section 3, we explain how to fit OCC into LL-SVM to model the coordinates for classification. We show our experimental results and comparison in Section 4, and conclude the paper in Section 5.

## 2 Anchor Plane Learning

In this section, we introduce our Orthogonal Coordinate Coding (OCC) based on some orthogonal basis vectors. For clarification, we summarize some notations in Table 1 which are used in LCC and OCC.

Table 1: *Some notations used in LCC and OCC.*

| Notation | Definition |
|---|---|
| $\mathbf{v} \in \mathbb{R}^d$ | A $d$-dimensional anchor point in LCC; a $d$-dimensional basis vector which defines a family of anchor planes in OCC. |
| $\mathcal{C} \subset \mathbb{R}^d$ | A subset in $d$-dimensional space containing all the anchor points ($\forall \mathbf{v}, \mathbf{v} \in \mathcal{C}$) in LCC; a subset in $d$-dimensional space containing all the basis vectors in OCC. |
| $\mathbf{C} \in \mathbb{R}^{d \times |\mathcal{C}|}$ | The anchor point (or basis vector) matrix with $\mathbf{v} \in \mathcal{C}$ as columns. |
| $\gamma_v(\mathbf{x}) \in \mathbb{R}$ | The local coding of a data point $\mathbf{x} \in \mathbb{R}^d$ using the anchor point (or basis vector) $\mathbf{v}$. |
| $\boldsymbol{\gamma}(\mathbf{x}) \in \mathbb{R}^d$ | The physical approximation vector of a data point $\mathbf{x}$. |
| $\boldsymbol{\gamma}_x \in \mathbb{R}^{|C|}$ | The coding vector of data point $\mathbf{x}$ containing all $\gamma_v(\mathbf{x})$ in order $\boldsymbol{\gamma}_x = [\gamma_v(\mathbf{x})]_{\mathbf{v} \in \mathcal{C}}$. |
| $\gamma$ | A map of $\mathbf{x} \in \mathbb{R}^d$ to $\boldsymbol{\gamma}_x$. |
| $(\gamma, \mathcal{C})$ | A coordinate coding. |

### 2.1 Preliminary

We first recall some definitions and lemmas in LCC based on which we develop our method. Notice that in the following sections, $\| \cdot \|$ denotes the $\ell_2$-norm without explicit explanation.

**Definition 1** (**Lipschitz Smoothness [18]**). *A function $f(\mathbf{x})$ on $\mathbb{R}^d$ is $(\alpha, \beta, p)$-Lipschitz smooth with respect to a norm $\| \cdot \|$ if $|f(\mathbf{x}') - f(\mathbf{x})| \leq \alpha \|\mathbf{x} - \mathbf{x}'\|$ and $|f(\mathbf{x}') - f(\mathbf{x}) - \nabla f(\mathbf{x})^T (\mathbf{x}' - \mathbf{x})| \leq \beta \|\mathbf{x} - \mathbf{x}'\|^{1+p}$, where we assume $\alpha, \beta > 0$ and $p \in (0, 1]$.*

**Definition 2** (**Coordinate Coding [18]**). *A coordinate coding is a pair $(\gamma, \mathcal{C})$, where $\mathcal{C} \subset \mathbb{R}^d$ is a set of anchor points, and $\gamma$ is a map of $\mathbf{x} \in \mathbb{R}^d$ to $[\gamma_v(\mathbf{x})]_{\mathbf{v} \in \mathcal{C}} \in \mathbb{R}^{|\mathcal{C}|}$ such that $\sum_{\mathbf{v}} \gamma_v(\mathbf{x}) = 1$. It induces the following physical approximation of $\mathbf{x}$ in $\mathbb{R}^d$: $\boldsymbol{\gamma}(\mathbf{x}) = \sum_{\mathbf{v} \in \mathcal{C}} \gamma_v(\mathbf{x})\mathbf{v}$. Moreover, for all $\mathbf{x} \in \mathbb{R}^d$, we define the corresponding coding norm as $\|\mathbf{x}\|_\gamma = (\sum_{\mathbf{v} \in \mathcal{C}} \gamma_v(\mathbf{x})^2)^{1/2}$.*

**Lemma 1** (**Linearization [18]**). *Let $(\gamma, \mathcal{C})$ be an arbitrary coordinate coding on $\mathbb{R}^d$. Let $f$ be an $(\alpha, \beta, p)$-Lipschitz smooth function. We have for all $\mathbf{x} \in \mathbb{R}^d$:*

$$\left| f(\mathbf{x}) - \sum_{\mathbf{v} \in \mathcal{C}} \gamma_v(\mathbf{x}) f(\mathbf{v}) \right| \leq \alpha \|\mathbf{x} - \boldsymbol{\gamma}(\mathbf{x})\| + \beta \sum_{\mathbf{v} \in \mathcal{C}} |\gamma_v(\mathbf{x})| \, \|\mathbf{v} - \boldsymbol{\gamma}(\mathbf{x})\|^{1+p} \quad (2)$$

As explained in [18], a good coding scheme for non-linear function approximation should make $\mathbf{x}$ close to its physical approximation $\boldsymbol{\gamma}(\mathbf{x})$ (*i.e.* smaller data reconstruction error $\|\mathbf{x} - \boldsymbol{\gamma}(\mathbf{x})\|$) and should be localized (*i.e.* smaller localization error $\sum_{\mathbf{v} \in \mathcal{C}} |\gamma_v(\mathbf{x})| \, \|\mathbf{v} - \boldsymbol{\gamma}(\mathbf{x})\|^{1+p}$). This is the basic idea of LCC.

**Definition 3** (**Localization Measure [18]**). *Given $\alpha$, $\beta$, $p$, and coding $(\gamma, \mathcal{C})$, we define*

$$Q_{\alpha, \beta, p}(\gamma, \mathcal{C}) = \mathbb{E}_x \left[ \alpha \|\mathbf{x} - \boldsymbol{\gamma}(\mathbf{x})\| + \beta \sum_{\mathbf{v} \in \mathcal{C}} |\gamma_v(\mathbf{x})| \, \|\mathbf{v} - \boldsymbol{\gamma}(\mathbf{x})\|^{1+p} \right] \quad (3)$$

Localization measure is equivalent to the expectation of the upper bound of the approximate error.

### 2.2 Orthogonal Coordinate Coding

In the following sections, we will follow the notations in Table 1, and define our orthogonal coordinate coding (OCC) as below.

**Definition 4** (**Orthogonal Coordinate Coding**). *An orthogonal coordinate coding is a pair $(\gamma, \mathcal{C})$, where $\mathcal{C} \subset \mathbb{R}^d$ contains $|\mathcal{C}|$ orthogonal basis vectors, that is, $\forall \mathbf{u}, \mathbf{v} \in \mathcal{C}$, if $\mathbf{u} \neq \mathbf{v}$, then $\mathbf{u}^T \mathbf{v} = 0$, and coding $\gamma$ is a map of $\mathbf{x} \in \mathbb{R}^d$ to $[\gamma_v(\mathbf{x})]_{\mathbf{v} \in \mathcal{C}} \in \mathbb{R}^{|\mathcal{C}|}$ such that $\gamma_v(\mathbf{x}) \propto \frac{\mathbf{x}^T \mathbf{v}}{\|\mathbf{v}\|^2}$ and $\sum_{\mathbf{v} \in \mathcal{C}} |\gamma_v(\mathbf{x})| = 1$.*

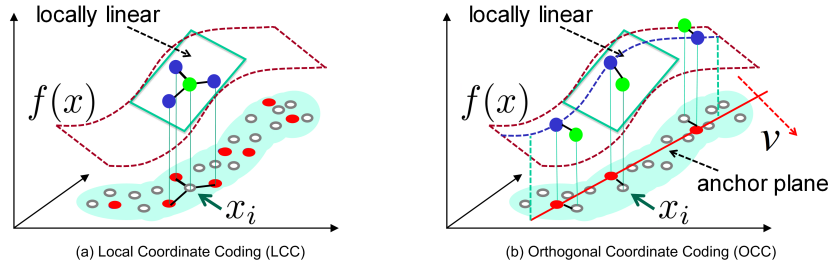

| (a) Local Coordinate Coding (LCC) | (b) Orthogonal Coordinate Coding (OCC) |

Figure 1: *Comparison of the geometric views on (a) LCC and (b) OCC, where the white and red dots denote the data and anchor points, respectively. In LCC, the anchor points are distributed among the data space and several nearest neighbors around the data are selected for data reconstruction, while in OCC the anchor points are located on the anchor plane defined by the normal vector (i.e. coordinate, basis vector)* $\mathbf{v}$ *and only the closest point to each data point on the anchor plane is selected for coding. The figures are borrowed from the slides of [17], and best viewed in color.*

Compared to Definition 2, there are two changes in OCC: (1) instead of anchor points we use a set of orthogonal basis vectors, which defines a set of anchor planes, and (2) the coding for each data point is defined on the $\ell_1$-norm unit ball, which removes the scaling factors in both $\mathbf{x}$ and $\mathbf{v}$. Notice that since given the data matrix, the maximum number of orthogonal basis vectors which can be used to represent all the data precisely is equal to the rank of the data matrix, the maximum value of $|\mathcal{C}|$ is equal to the rank of the data matrix as well.

Figure 1 illustrates the geometric views on LCC and OCC respectively. Intuitively, in both methods anchor points try to encode data locally. However, the ways of their arrangement are quite different. In LCC anchor points are distributed among the whole data space such that each data can be covered by certain anchor points in a local region, and their distribution cannot be described using regular shapes. On the contrary, the anchor points in OCC are located on the anchor plane defined by a basis vector. In fact, each anchor plane can be considered as infinite number of anchor points, and for each data point only its closest point on each anchor plane is utilized for reconstruction. Therefore, intuitively the number of anchor planes in OCC should be much fewer than the number of anchor points in LCC.

**Theorem 1** (**Localization Error of OCC**). *Let $(\gamma, \mathcal{C})$ be an orthogonal coordinate coding on $\mathbb{R}^d$ where $\mathcal{C} \subset \mathbb{R}^d$ with size $|\mathcal{C}| = M$. Let $f$ be an $(\alpha, \beta, p)$-Lipschitz smooth function. Without losing generalization, assuming $\forall \mathbf{x} \in \mathbb{R}^d, \|\mathbf{x}\| \leq 1$ and $\forall \mathbf{v} \in \mathcal{C}, 1 \leq \|\mathbf{v}\| \leq h(h \geq 1)$, then the localization error in Lemma 1 is bounded by:*

$$\sum_{\mathbf{v} \in \mathcal{C}} |\gamma_v(\mathbf{x})| \, \|\mathbf{v} - \boldsymbol{\gamma}(\mathbf{x})\|^{1+p} \leq \left[(1+M)h\right]^{1+p}. \tag{4}$$

*Proof.* Let $\gamma_v(\mathbf{x}) = \frac{\mathbf{x}^T \mathbf{v}}{s_x \|\mathbf{v}\|^2}$, where $s_x = \sum_{\mathbf{v} \in \mathcal{C}} \frac{|\mathbf{x}^T \mathbf{v}|}{\|\mathbf{v}\|^2}$, then

$$\sum_{\mathbf{v} \in \mathcal{C}} |\gamma_v(\mathbf{x})| \, \|\mathbf{v} - \boldsymbol{\gamma}(\mathbf{x})\|^{1+p} = \sum_{\mathbf{v} \in \mathcal{C}} |\gamma_v(\mathbf{x})| \left[ \|\mathbf{v}\|^2 - 2s_x \gamma_v(\mathbf{x}) \|\mathbf{v}\|^2 + \sum_{\mathbf{v} \in \mathcal{C}} s_x^2 \gamma_v(\mathbf{x})^2 \|\mathbf{v}\|^2 \right]^{\frac{1+p}{2}}$$

$$\leq \sum_{\mathbf{v} \in \mathcal{C}} |\gamma_v(\mathbf{x})| \left[ \|\mathbf{v}\|^2 + 2s_x \|\mathbf{v}\|^2 |\gamma_v(\mathbf{x})| + \sum_{\mathbf{v} \in \mathcal{C}} s_x^2 \gamma_v(\mathbf{x})^2 \|\mathbf{v}\|^2 \right]^{\frac{1+p}{2}}$$

$$\leq \sum_{\mathbf{v} \in \mathcal{C}} |\gamma_v(\mathbf{x})| \left[ \|\mathbf{v}\|^2 + 2s_x \|\mathbf{v}\|^2 |\gamma_v(\mathbf{x})| + \left( \sum_{\mathbf{v} \in \mathcal{C}} \gamma_v(\mathbf{x})^2 \right) \left( \max_{\mathbf{v} \in \mathcal{C}} s_x^2 \|\mathbf{v}\|^2 \right) \right]^{\frac{1+p}{2}} \tag{5}$$

$\because \forall \mathbf{x} \in \mathbb{R}^d, \|\mathbf{x}\| \leq 1$ and $\forall \mathbf{v} \in \mathcal{C}, 1 \leq \|\mathbf{v}\| \leq h(h \geq 1), \sum_{\mathbf{v} \in \mathcal{C}} |\gamma_v(\mathbf{x})| = 1$,
$\therefore \forall \mathbf{v} \in \mathcal{C}, |\gamma_v(\mathbf{x})| \leq 1, \sum_{\mathbf{v} \in \mathcal{C}} \gamma_v(\mathbf{x})^2 \leq 1, s_x = \sum_{\mathbf{v} \in \mathcal{C}} \frac{|\mathbf{x}^T \mathbf{v}|}{\|\mathbf{v}\|^2} \leq \sum_{\mathbf{v} \in \mathcal{C}} \frac{\|\mathbf{x}\| \|\mathbf{v}\|}{\|\mathbf{v}\|^2} \leq M$

$$\therefore \quad \sum_{\mathbf{v}\in\mathcal{C}} |\gamma_v(\mathbf{x})|\, \|\mathbf{v} - \boldsymbol{\gamma}(\mathbf{x})\|^{1+p} \quad \leq \quad \sum_{\mathbf{v}\in\mathcal{C}} |\gamma_v(\mathbf{x})| \left[ h^2 + 2Mh^2|\gamma_v(\mathbf{x})| + M^2h^2 \right]^{\frac{1+p}{2}}$$

$$= \quad h^{1+p} \sum_{\mathbf{v}\in\mathcal{C}} |\gamma_v(\mathbf{x})| \left[ 1 + 2M|\gamma_v(\mathbf{x})| + M^2 \right]^{\frac{1+p}{2}}$$

$$\leq \quad h^{1+p} \left( \sum_{\mathbf{v}\in\mathcal{C}} |\gamma_v(\mathbf{x})| \right) \left( \max_{\mathbf{v}\in\mathcal{C}} \left\{ \left[ 1 + 2M|\gamma_v(\mathbf{x})| + M^2 \right]^{\frac{1+p}{2}} \right\} \right)$$

$$= \quad h^{1+p} \cdot \max_{\mathbf{v}\in\mathcal{C}} \left\{ \left[ 1 + 2M|\gamma_v(\mathbf{x})| + M^2 \right]^{\frac{1+p}{2}} \right\}$$

$$\leq \quad h^{1+p} \left[ 1 + 2M + M^2 \right]^{\frac{1+p}{2}} = \left[ (1+M)h \right]^{1+p}. \tag{6}$$

$\square$

## 2.3 Learning Orthogonal Basis Vectors

Instead of optimizing Definition 3, LCC simplifies the localization error term by assuming $\boldsymbol{\gamma}(\mathbf{x}) = \mathbf{x}$ and $p = 1$. Mathematically LCC solves the following optimization problem:

$$\min_{(\gamma,\mathcal{C})} \sum_{\mathbf{x}\in\mathcal{X}} \left\{ \frac{1}{2}\|\mathbf{x} - \boldsymbol{\gamma}(\mathbf{x})\|^2 + \mu \sum_{\mathbf{v}\in\mathcal{C}} |\gamma_v(\mathbf{x})|\|\mathbf{v} - \mathbf{x}\|^2 + \lambda \sum_{\mathbf{v}\in\mathcal{C}} \|\mathbf{v}\|^2 \right\} \tag{7}$$
$$\text{s.t.} \quad \forall \mathbf{x}, \quad \sum_{\mathbf{v}\in\mathcal{C}} \gamma_v(\mathbf{x}) = 1.$$

They update $\mathcal{C}$ and $\gamma$ via alternating optimization. The step of updating $\gamma$ can be transformed into a canonical LASSO problem, and the step of updating $\mathcal{C}$ is a least squares problem.

For OCC, given an $(\alpha, \beta, p)$-Lipschitz smooth function $f$ and a set of data $\mathcal{X} \subset \mathbb{R}^d$, whose corresponding data matrix and its rank are denoted as $\mathbf{X}$ and $D$, respectively, we would like to learn an orthogonal coordinate coding $(\gamma, \mathcal{C})$ where the number of basis vectors $|\mathcal{C}| = M \leq D$ such that the localization measure of this coding is minimized. Since Theorem 1 proves that the localization error per data point is bounded by a constant given an OCC, in practice we only need to minimize the data reconstruction error in order to minimize the upper bound of the localization measure. That is, we need to solve the following problem:

$$\min_{(\gamma,\mathcal{C})} \sum_{\mathbf{x}\in\mathcal{X}} \|\mathbf{x} - \mathbf{C}\boldsymbol{\gamma}_x\|^2 \tag{8}$$
$$\text{s.t.} \quad \forall \mathbf{u}, \mathbf{v} \in \mathcal{C}, \quad \mathbf{u} \neq \mathbf{v} \Rightarrow \mathbf{u}^T\mathbf{v} = 0,$$
$$|\mathcal{C}| = M,$$
$$\forall \mathbf{x}, \quad \|\boldsymbol{\gamma}_x\|_1 = 1.$$

This optimization problem is quite similar to sparse coding [12], except that there exists the orthogonal constraint on the basis vectors. In practice we relax this problem by removing the constraint $\forall \mathbf{x}, \|\boldsymbol{\gamma}_x\|_1 = 1$.

**(I) Solving for $\mathcal{C}$.** Eqn. 8 can be solved first using singular value decomposition (SVD). Let the SVD of $\mathbf{X} = \mathbf{V}\boldsymbol{\Sigma}\mathbf{U}$ where the singular values are positive and in descending order with respect to $\boldsymbol{\Sigma}$. Then we set $\mathbf{C} = \mathbf{V}^{\{d\times M\}}\boldsymbol{\Sigma}^{\{M\times M\}}$, where $\mathbf{V}^{\{d\times M\}}$ denotes a sub-matrix of $\mathbf{V}$ containing the elements within rows from 1 to $d$ and columns from 1 to $M$, similarly for $\boldsymbol{\Sigma}^{\{M\times M\}}$. We need only to use a few top eigenvectors as our orthogonal basis vectors for coding, and the search space is far smaller than generating anchor points.

**(II) Solving for $\boldsymbol{\gamma}_x$.** Since we have the orthogonal basis vectors in $\mathcal{C}$, we can easily derive the formulation for calculating $\tilde{\boldsymbol{\gamma}}_x$, the value of $\boldsymbol{\gamma}_\mathbf{x}$ before normalization, that is, $\tilde{\boldsymbol{\gamma}}_x = (\mathbf{C}^T\mathbf{C})^{-1}\mathbf{C}^T\mathbf{x}$. Letting $\{\bar{\mathbf{v}}\}$ and $\{\sigma_v\}$ be the corresponding singular vectors and singular values, based on the orthogonality of basis vectors we have $\tilde{\gamma}_v(\mathbf{x}) = \frac{\bar{\mathbf{v}}^T\mathbf{x}}{\sigma_v}$, which is a variant of the coding definition in Definition 4. Finally, we can calculate $\boldsymbol{\gamma}_x$ by normalizing $\tilde{\boldsymbol{\gamma}}_x$ as follows: $\boldsymbol{\gamma}_x = \frac{\tilde{\boldsymbol{\gamma}}_x}{\|\tilde{\boldsymbol{\gamma}}_x\|_1}$.

## 3 Modeling Classification Decision Boundary in SVM

Given a set of data $\{(\mathbf{x}_i, y_i)\}$ where $y_i \in \{-1, 1\}$ is the label of $\mathbf{x}_i$, the decision boundary for binary classification of a linear SVM is $f(\mathbf{x}) = \mathbf{w}^T \mathbf{x} + b$ where $\mathbf{w}$ is the normal vector of the decision hyperplane (*i.e.* coefficients) of the SVM and $b$ is a bias term. Here, we assume that the decision boundary is an $(\alpha, \beta, p)$-Lipschitz smooth function. Since in LCC each data is encoded by some anchor points on the data manifold, it can model the decision boundary of an SVM directly using $f(\mathbf{x}) \approx \sum_{\mathbf{v} \in \mathcal{C}} \gamma_v(\mathbf{x}) f(\mathbf{v})$. Then by taking $\boldsymbol{\gamma}_x$ as the input data of a linear SVM, $f(\mathbf{v})$'s can be learned to approximate the decision boundary $f$.

However, OCC learns a set of orthogonal basis vectors, rather than anchor points, and corresponding coding for data. This makes OCC suitable to model the normal vectors of decision hyperplanes in SVMs locally with LL-SVM. Given data $\mathbf{x}$ and an orthogonal coordinate coding $(\gamma, \mathcal{C})$, the decision boundary in LL-SVM can be formulated as follows [1].

$$f(\mathbf{x}) = \mathbf{w}(\mathbf{x})^T \mathbf{x} + b = \sum_{\mathbf{v} \in \mathcal{C}} \gamma_v(\mathbf{x}) \mathbf{w}(\mathbf{v})^T \mathbf{x} + b = \boldsymbol{\gamma}_x^T \mathbf{W} \mathbf{x} + b \qquad (9)$$

where $\mathbf{W} \in \mathbb{R}^{M \times d}$ is a matrix which needs to be learned for SVMs. In the view of kernel SVMs, we actually define another kernel $\mathbf{K}$ based on $\mathbf{x}$ and $\boldsymbol{\gamma}_x$ as shown below.

$$\forall i, j, \ \mathbf{K}(\mathbf{x}_i, \mathbf{x}_j) = <\boldsymbol{\gamma}_{x_i} \mathbf{x}_i^T, \boldsymbol{\gamma}_{x_j} \mathbf{x}_j^T> \qquad (10)$$

where $< \cdot, \cdot >$ denotes the Frobenius inner product. Notice that in our kernel, *latent semantic kernel* [6] has been involved which is defined based on a set of orthogonal basis vectors.

## 4 Experiments

In our experiments, we test OCC with LL-SVM for classification on the benchmark datasets: MNIST, USPS and LETTER. The features we used are the raw features such that we can compare our results fairly with others.

MNIST contains 40000 training and 10000 test gray-scale images with resolution $28 \times 28$, which are normalized directly into 784 dimensional vectors. The label of each image is one of the 10 digits from 0 to 9. USPS contains 7291 training and 2007 test gray-scale images with resolution 16 x 16, directly stored as 256 dimensional vectors, and the label of each image still corresponds to one of the 10 digits from 0 to 9. LETTER contains 16000 training and 4000 testing images, each of which is represented as a relatively short 16 dimensional vector, and the label of each image corresponds to one of the 26 letters from A to Z.

We re-implemented LL-SVM based on the C++ code of LIBLINEAR [8] [2] and PEGASOS [14] [3], respectively, and performed multi-class classification using the one-vs-all strategy. This aims to test the effect of either quadratic programming or stochastic gradient based SVM solver on both accuracy and computational time. We denote these two ways of LL-SVM as *LIB-LLSVM* and *PEG-LLSVM* for short. We tried to learn our basis vectors in two ways: (1) SVD is applied directly to the entire training data matrix, or (2) SVD is applied separately to the data matrix consisting of all the positive training data. We denote these two types of OCC as *G-OCC* (*i.e.* Generic OCC) and *C-OCC* (*i.e.* Class-specific OCC), respectively. Then the coding for each data is calculated as explained in Section 2.3. Next, all the training raw features and their coding vectors are taken as the input to train the model $(\mathbf{W}, b)$ of LL-SVM. For each test data $\mathbf{x}$, we calculate its coding in the same way and classify it based on its decision values, that is, $y(\mathbf{x}) = \arg\max_y \boldsymbol{\gamma}_{x,y}^T \mathbf{W}_y \mathbf{x} + b_y$.

Figure 2 shows the comparison of classification error rates among *G-OCC + LIB-LLSVM*, *G-OCC + PEG-LLSVM*, *C-OCC + LIB-LLSVM*, and *C-OCC + PEG-LLSVM* on MNIST (left), USPS (middle), and LETTER (right), respectively, using different numbers of orthogonal basis vectors. With the same OCC, LIB-LLSVM performs slightly better than PEG-LLSVM in terms of accuracy, and both

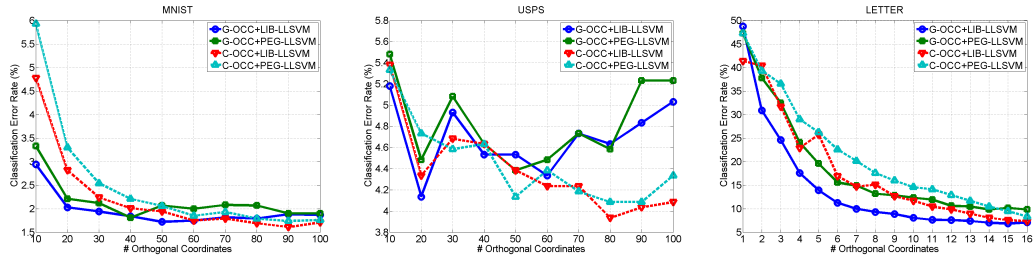

Figure 2: *Performance comparison among the 4 different combinations of OCC + LL-SVM on MNIST (left), USPS (middle), and LETTER (right) using different numbers of orthogonal basis vectors. This figure is best viewed in color.*

behaves similarly with the increase of the number of orthogonal basis vectors. It seems that in general C-OCC is better than G-OCC.

Table 2 summarizes our comparison results between our methods and some other SVM based approaches. The parameters of the RBF kernel used in the kernel SVMs are the same as [2]. Since there are no results of LCC on USPS and LETTER or its code, we tested the published code of LLC [16] on these two datasets so that we can have a rough idea of how well LCC works. The anchor points are found using K-Means. From Table 2, we can see that applying linear SVM directly on OCC works slightly better than on the raw features, and when OCC is working with LL-SVM, the performance is boosted significantly while the numbers of anchor points that are needed in LL-SVM are reduced. On MNIST we can see that our non-linear function approximation is better than LCC, improved LCC, LLC, and LL-SVM, on USPS ours is better than both LLC and LL-SVM, but on LETTER ours is worse than LLC (4096 anchor points) and LL-SVM (100 anchor points). The reason for this is that strictly speaking LETTER is not a high dimensional dataset (only 16 dimensions per data), which limits the power of OCC. Compared with kernel based SVMs, our method can achieve comparable or even better results (*e.g.* on USPS). All of these results demonstrate that OCC is quite suitable to model the non-linear normal vectors using linear SVMs for classification on high dimensional data. In summary, our encoding scheme uses much less number of basis vectors compared to anchor points in LCC while achieving better test accuracy, which translates to higher performance both in terms of generalization and efficiency in computation.

We show our training and test time on these three datasets as well in Table 3 based on unoptimized MATLAB code on a single thread of a 2.67 GHz CPU. For training, the time includes calculating OCC and training LL-SVM. From this table, we can see that our methods are a little slower than the original LL-SVM, but still much faster than kernel SVMs. The main reason for this is that OCC is non-sparse while in [11] the coefficients are sparse. However, for calculating coefficients, OCC is faster than [11], because there is no distance calculation or $K$ nearest neighbor search involved in OCC, just simple multiplication and normalization.

## 5    Conclusion

In this paper, we propose orthogonal coordinate coding (OCC) to encode high dimensional data based on a set of anchor planes defined by a set of orthogonal basis vectors. Theoretically we prove that our OCC can guarantee a fixed upper bound of approximation error for any $(\alpha, \beta, p)$-Lipschitz smooth function, and we can easily learn the orthogonal basis vectors using SVD to minimize the localization measure. Meanwhile, OCC can help locally linear SVM (LL-SVM) approximate the kernel-based SVMs, and our experiments demonstrate that with a few orthogonal anchor planes, LL-SVM can achieve comparable or better results than LCC and its variants improved LCC and LLC with linear SVMs, and on USPS even better than kernel-based SVMs. In future, we would like to learn the orthogonal basis vectors using semi-definite programming to guarantee the orthogonality.

**Acknowledgements**. We thank J. Valentin, P. Sturgess and S. Sengupta for useful discussion in this paper. This work was supported by the IST Programme of the European Community, under

Table 2: *Classification error rate comparison (%) between our methods and others on MNIST, USPS, and LETTER. The numbers of anchor planes in the brackets are the ones which returns the best result on each dataset. All kernel methods [13, 14, 15, 16, 17] use the RBF kernel. In general, LIB-LLSVM + C-OCC performs best.*

| Methods | MNIST | USPS | LETTER |
|---|---|---|---|
| Linear SVM + G-OCC (# basis vectors) | 9.25 (100) | 7.82 (95) | 30.52 (15) |
| Linear SVM + C-OCC (# anchor planes) | 7.42 (100) | 5.98 (95) | 14.95 (16) |
| LIB-LLSVM + G-OCC (# basis vectors) | 1.72 (50) | 4.14 (20) | 6.85 (15) |
| PEG-LLSVM + G-OCC (# basis vectors) | 1.81 (40) | 4.38 (50) | 9.83 (14) |
| LIB-LLSVM + C-OCC (# basis vectors) | 1.61 (90) | **3.94 (80)** | 7.35 (16) |
| PEG-LLSVM + C-OCC (# basis vectors) | 1.74 (90) | 4.09 (80) | 8.30 (16) |
| Linear SVM (10 passes) [1] | 12.00 | 9.57 | 41.77 |
| Linear SVM + LCC (512 anchor points) [18] | 2.64 | - | - |
| Linear SVM + LCC (4096 anchor points) [18] | 1.90 | - | - |
| Linear SVM + improved LCC (512 anchor points) [19] | 1.95 | - | - |
| Linear SVM + improved LCC (4096 anchor points) [19] | 1.64 | - | - |
| Linear SVM + LLC (512 anchor points) [16] | 3.69 | 5.78 | 9.02 |
| Linear SVM + LLC (4096 anchor points) [16] | 2.28 | 4.38 | 4.12 |
| LibSVM [4] | **1.36** | - | - |
| LA-SVM (1 pass) [3] | 1.42 | - | - |
| LA-SVM (2 passes) [3] | **1.36** | - | - |
| MCSVM [5] | 1.44 | 4.24 | 2.42 |
| $SVM_{struct}$[15] | 1.40 | 4.38 | **2.40** |
| LA-RANK (1 pass) [2] | 1.41 | 4.25 | 2.80 |
| LL-SVM (100 anchor points, 10 passes) [11] | 1.85 | 5.78 | 5.32 |

Table 3: *Computational time comparison between our methods and others on MNIST, USPS, and LETTER. The numbers in Row 7-14 are copied from [11]. The training times of our methods include the calculation of OCC and training LL-SVM. All the numbers are corresponding to the methods shown in Table 2 with the same parameters. Notice that for PEG-LLSVM, $10^6$ random data points is used for training.*

| Methods | Training Time (s) | | | Test Time (ms) | | |
|---|---|---|---|---|---|---|
| | MNIST | USPS | LETTER | MNIST | USPS | LETTER |
| LIB-LLSVM + G-OCC | 113.38 | 5.78 | 4.14 | $5.51\times10^3$ | 19.23 | 4.09 |
| PEG-LLSVM + G-OCC | 125.03 | 14.50 | 2.02 | 302.28 | 23.25 | 3.33 |
| LIB-LLSVM + C-OCC | 224.09 | 25.61 | 1.66 | $9.57\times10^3$ | 547.60 | 63.13 |
| PEG-LLSVM + C-OCC | 273.70 | 23.31 | 0.85 | 503.18 | 50.63 | 28.94 |
| Linear SVM (10 passes) [1] | 1.5 | 0.26 | 0.18 | $8.75\times10^{-3}$ | - | - |
| LibSVM [4] | $1.75\times10^4$ | - | - | 46 | - | - |
| LA-SVM (1 pass) [3] | $4.9\times10^3$ | - | - | 40.6 | - | - |
| LA-SVM (2 passes) [3] | $1.22\times10^4$ | - | - | 42.8 | - | - |
| MCSVM [5] | $2.5\times10^4$ | 60 | $1.2\times10^3$ | - | - | - |
| $SVM_{struct}$[15] | $2.65\times10^5$ | $6.3\times10^3$ | $2.4\times10^4$ | - | - | - |
| LA-RANK (1 pass) [2] | $3\times10^4$ | 85 | 940 | - | - | - |
| LL-SVM (100, 10 passes) [11] | 81.7 | 6.2 | 4.2 | 0.47 | - | - |

the PASCAL2 Network of Excellence, IST-2007-216886. P. H. S. Torr is in receipt of Royal Society Wolfson Research Merit Award.

**References**

[1] Bordes, A., Bottou, L. & Gallinari, P. (2009) Sgd-qn: Careful quasi-newton stochastic gradient descent. *Journal of Machine Learning Research (JMLR).*

[2] Bordes, A., Bottou, L., Gallinari, P., & Weston, J. (2007) Solving multiclass support vector machines with larank. In *Proceeding of International Conference on Machine Learning (ICML).*

[3] Bordes, A., Ertekin, S., Weston, J., & Bottou, L. (2005) Fast kernel classifiers with online and active learning. *Journal of Machine Learning Research (JMLR).*

[4] Chang, C. & Lin, C. (2011) LIBSVM: A Library for Support Vector Machines. *ACM Transactions on Intelligent Systems and Technology*, vol. 2, issue 3, pp. 27:1-27:27.

[5] Crammer, K. & Singer, Y. (2002) On the algorithmic implementation of multiclass kernel-based vector machines. *Journal of Machine Learning Research (JMLR).*

[6] Cristianini, N., Shawe-Taylor, J. & Lodhi, H. (2002) Latent Semantic Kernels. *Journal of Intelligent Information Systems*, Vol. 18, No. 2-3, 127-152.

[7] Everingham, M., Van Gool, L., Williams, C.K.I., Winn, J. & Zisserman, A. The PASCAL Visual Object Classes Challenge 2009 (VOC2009). *http://www.pascal-network.org/challenges/VOC/voc2009/workshop/index.html*

[8] Fan, R., Chang, K., Hsieh, C., Wang, X. & Lin, C. (2008) LIBLINEAR: A Library for Large Linear Classification. *Journal of Machine Learning Research (JMLR)*, vol. 9, pp. 1871-1874.

[9] Gönen, M. & Alpaydin, E. (2008) Localized Multiple Kernel Learning. In *Proceeding of International Conference on Machine Learning (ICML).*

[10] Kecman, V. & Brooks, J.P. (2010) Locally Linear Support Vector Machines and Other Local Models. In *Proceeding of IEEE World Congress on Computational Intelligence (WCCI)*, pp. 2615-2620.

[11] Ladicky, L. & Torr, P.H.S. (2011) Locally Linear Support Vector Machines. In *Proceeding of International Conference on Machine Learning (ICML).*

[12] Lee, H., Battle, A., Raina, R., & Ng, A.Y. (2007) Efficient Sparse Coding Algorithms. In *Advances in Neural Information Processing Systems (NIPS).*

[13] Mairal, J., Bach, F., Ponce, J. & Sapiro, G. (2009) Online Dictionary Learning for Sparse Coding. In *Proceeding of International Conference on Machine Learning (ICML).*

[14] Shalev-Shwartz, S., Singer, Y., & Srebro, N. (2007) Pegasos: Primal Estimated sub-GrAdient SOlver for SVM. In *Proceeding of International Conference on Machine Learning (ICML).*

[15] Tsochantaridis, I., Joachims, T., Hofmann, T., & Altun, Y. (2005) Large margin methods for structured and interdependent output variables. *Journal of Machine Learning Research (JMLR).*

[16] Wang, J., Yang, J., Yu, K., Lv, F., Huang, T., & Gong, Y. (2010) Locality-constrained Linear Coding for Image Classification. In *Proceedings of IEEE Conference on Computer Vision and Pattern Recognition (CVPR).*

[17] Yu, K. & Ng, A. (2010) ECCV-2010 Tutorial: Feature Learning for Image Classification. *http://ufldl.stanford.edu/eccv10-tutorial/.*

[18] Yu, K., Zhang, T., & Gong, Y. (2009) Nonlinear Learning using Local Coordinate Coding. In *Advances in Neural Information Processing Systems (NIPS).*

[19] Yu, K. & Zhang, T. (2010) Improved Local Coordinate Coding using Local Tangents. In *Proceeding of International Conference on Machine Learning (ICML).*

[20] Zhang, H., Berg, A., Maure, M. & Malik, J. (2006) SVM-KNN: Discriminative nearest neighbor classification for visual category recognition. In *Proceedings of IEEE Conference on Computer Vision and Pattern Recognition (CVPR)*, pp. 2126-2136.

## Footnotes

[1]Notice that Eqn. 9 is slightly different from the original formulation in [11] by ignoring the different bias term for each orthogonal basis vector.

[2]Using LIBLINEAR, we implemented LL-SVM based on Eqn. 9.

[3]Using PEGASOS, we implemented LL-SVM based on the original formulation in [11].
